# Sparse Code Shrinkage: Denoising by Nonlinear Maximum Likelihood Estimation

**Aapo Hyvärinen, Patrik Hoyer and Erkki Oja**
Helsinki University of Technology
Laboratory of Computer and Information Science
P.O. Box 5400, FIN-02015 HUT, Finland
`aapo.hyvarinen@hut.fi,patrik.hoyer@hut.fi,erkki.oja@hut.fi`
`http://www.cis.hut.fi/projects/ica/`

## Abstract

Sparse coding is a method for finding a representation of data in which each of the components of the representation is only rarely significantly active. Such a representation is closely related to redundancy reduction and independent component analysis, and has some neurophysiological plausibility. In this paper, we show how sparse coding can be used for denoising. Using maximum likelihood estimation of nongaussian variables corrupted by gaussian noise, we show how to apply a shrinkage nonlinearity on the components of sparse coding so as to reduce noise. Furthermore, we show how to choose the optimal sparse coding basis for denoising. Our method is closely related to the method of wavelet shrinkage, but has the important benefit over wavelet methods that both the features and the shrinkage parameters are estimated directly from the data.

## 1 Introduction

A fundamental problem in neural network research is to find a suitable representation for the data. One of the simplest methods is to use linear transformations of the observed data. Denote by $\mathbf{x} = (x_1, x_2, ..., x_n)^T$ the observed $n$-dimensional random vector that is the input data (e.g., an image window), and by $\mathbf{s} = (s_1, s_2, ..., s_n)^T$ the vector of the linearly transformed component variables. Denoting further the $n \times n$ transformation matrix by $\mathbf{W}$, the linear representation is given by

$$\mathbf{s} = \mathbf{W}\mathbf{x}. \tag{1}$$

We assume here that the number of transformed components equals the number of observed variables, but this need not be the case in general.

An important representation method is given by (linear) sparse coding [1, 10], in which the representation of the form (1) has the property that only a small number of the components $s_i$ of the representation are significantly non-zero at the same time. Equivalently, this means that a given component has a 'sparse' distribution. A random variable $s_i$ is called sparse when $s_i$ has a distribution with a peak at zero, and heavy tails, as is the case, for example, with the double exponential (or Laplace) distribution [6]; for all practical purposes, sparsity is equivalent to supergaussianity or leptokurtosis [8]. Sparse coding is an adaptive method, meaning that the matrix **W** is estimated for a given class of data so that the components $s_i$ are as sparse as possible; such an estimation procedure is closely related to independent component analysis [2].

Sparse coding of sensory data has been shown to have advantages from both physiological and information processing viewpoints [1]. However, thorough analyses of the utility of such a coding scheme have been few. In this paper, we introduce and analyze a statistical method based on sparse coding. Given a signal corrupted by additive gaussian noise, we attempt to *reduce gaussian noise* by soft thresholding ('shrinkage') of the sparse components. Intuitively, because only a few of the components are significantly active in the sparse code of a given data point, one may assume that the activities of components with small absolute values are purely noise and set them to zero, retaining just a few components with large activities. This method is closely connected to the wavelet shrinkage method [3]. In fact, sparse coding may be viewed as a principled way for determining a wavelet-like basis and the corresponding shrinkage nonlinearities, based on data alone.

## 2    Maximum likelihood estimation of sparse components

The starting point of a rigorous derivation of our denoising method is the fact that the distributions of the sparse components are nongaussian. Therefore, we shall begin by developing a general theory that shows how to remove gaussian noise from nongaussian variables, making minimal assumptions on the data.

Denote by $s$ the original nongaussian random variable (corresponding here to a noise-free version of one of the sparse components $s_i$), and by $\nu$ gaussian noise of zero mean and variance $\sigma^2$. Assume that we only observe the random variable $y$:

$$y = s + \nu \tag{2}$$

and we want to estimate the original $s$. Denoting by $p$ the probability density of $s$, and by $f = -\log p$ its negative log-density, the maximum likelihood (ML) method gives the following estimator for $s$:

$$\hat{s} = \arg\min_u \frac{1}{2\sigma^2}(y - u)^2 + f(u). \tag{3}$$

Assuming $f$ to be strictly convex and differentiable, this can be solved [6] to yield $\hat{s} = g(y)$, where the function $g$ can be obtained from the relation

$$g^{-1}(u) = u + \sigma^2 f'(u). \tag{4}$$

This nonlinear estimator forms the basis of our method.

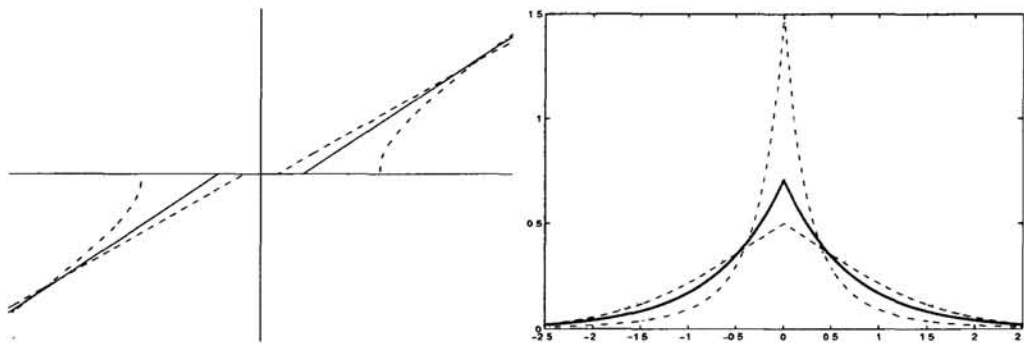

Figure 1: Shrinkage nonlinearities and associated probability densities. Left: Plots of the different shrinkage functions. Solid line: shrinkage corresponding to Laplace density. Dashed line: typical shrinkage function obtained from (6). Dash-dotted line: typical shrinkage function obtained from (8). For comparison, the line $x = y$ is given by dotted line. All the densities were normalized to unit variance, and noise variance was fixed to .3. Right: Plots of corresponding model densities of the sparse components. Solid line: Laplace density. Dashed line: a typical moderately super-gaussian density given by (5). Dash-dotted line: a typical strongly supergaussian density given by (7). For comparison, gaussian density is given by dotted line.

## 3   Parameterizations of sparse densities

To use the estimator defined by (3) in practice, the densities of the $s_i$ need to be modelled with a parameterization that is rich enough. We have developed two parameterizations that seem to describe very well most of the densities encountered in image denoising. Moreover, the parameters are easy to estimate, and the inversion in (4) can be performed analytically. Both models use two parameters and are thus able to model different degrees of supergaussianity, in addition to different scales, i.e. variances. The densities are here assumed to be symmetric and of zero mean.

The *first model* is suitable for supergaussian densities that are not sparser than the Laplace distribution [6], and is given by the family of densities

$$p(s) = C \exp(-as^2/2 - b|s|), \tag{5}$$

where $a, b > 0$ are parameters to be estimated, and $C$ is an irrelevant scaling constant. The classical Laplace density is obtained when $a = 0$, and gaussian densities correspond to $b = 0$. A simple method for estimating $a$ and $b$ was given in [6]. For this density, the nonlinearity $g$ takes the form:

$$g(u) = \frac{1}{1 + \sigma^2 a} \text{sign}(u) \max(0, |u| - b\sigma^2) \tag{6}$$

where $\sigma^2$ is the noise variance. The effect of the *shrinkage* function in (6) is to reduce the absolute value of its argument by a certain amount, which depends on the parameters, and then rescale. Small arguments are thus set to zero. Examples of the obtained shrinkage functions are given in Fig. 1.

The *second model* describes densities that are sparser than the Laplace density:

$$p(s) = \frac{1}{2d} \frac{(\alpha + 2) \left[\alpha (\alpha + 1)/2\right]^{(\alpha/2+1)}}{[\sqrt{\alpha (\alpha + 1)/2} + |s/d|]^{(\alpha+3)}}. \tag{7}$$

When $\alpha \to \infty$, the Laplace density is obtained as the limit. A simple consistent method for estimating the parameters $d, \alpha > 0$ in (7) can be obtained from the relations $d = \sqrt{E\{s^2\}}$ and $\alpha = (2 - k + \sqrt{k(k + 4)})/(2k - 1)$ with $k = d^2 p_s(0)^2$, see [6]. The resulting shrinkage function can be obtained as [6]

$$g(u) = \text{sign}(u) \max(0, \frac{|u| - ad}{2} + \frac{1}{2}\sqrt{(|u| + ad)^2 - 4\sigma^2(\alpha + 3)}) \qquad (8)$$

where $a = \sqrt{\alpha(\alpha + 1)/2}$, and $g(u)$ is set to zero in case the square root in (8) is imaginary. This is a shrinkage function that has a certain hard-thresholding flavor, as depicted in Fig. 1.

Examples of the shapes of the densities given by (5) and (7) are given in Fig. 1, together with a Laplace density and a gaussian density. For illustration purposes, the densities in the plot are normalized to unit variance, but these parameterizations allow the variance to be choosen freely.

Choosing whether model (5) or (7) should be used can be based on moments of the distributions; see [6]. Methods for estimating the noise variance $\sigma^2$ are given in [3, 6].

## 4 Sparse code shrinkage

The above results imply the following *sparse code shrinkage* method for denoising. Assume that we observe a noisy version $\tilde{\mathbf{x}} = \mathbf{x} + \boldsymbol{\nu}$ of the data $\mathbf{x}$, where $\boldsymbol{\nu}$ is gaussian white noise vector. To denoise $\tilde{\mathbf{x}}$, we transform the data to a sparse code, apply the above ML estimation procedure component-wise, and then transform back to the original variables. Here, we constrain the transformation to be orthogonal; this is motivated in Section 5. To summarize:

1. First, using a noise-free training set of $\mathbf{x}$, use some sparse coding method for determining the orthogonal matrix $\mathbf{W}$ so that the components $s_i$ in $\mathbf{s} = \mathbf{W}\mathbf{x}$ have as sparse distributions as possible. Estimate a density model $p_i(s_i)$ for each sparse component, using the models in (5) and (7).

2. Compute for each noisy observation $\tilde{\mathbf{x}}(t)$ of $\mathbf{x}$ the corresponding noisy sparse components $\mathbf{y}(t) = \mathbf{W}\tilde{\mathbf{x}}(t)$. Apply the shrinkage non-linearity $g_i(.)$ as defined in (6), or in (8), on each component $y_i(t)$, for every observation index $t$. Denote the obtained components by $\hat{s}_i(t) = g_i(y_i(t))$.

3. Invert the relation (1) to obtain estimates of the noise-free $\mathbf{x}$, given by $\hat{\mathbf{x}}(t) = \mathbf{W}^T \hat{\mathbf{s}}(t)$.

To estimate the sparsifying transform $\mathbf{W}$, we assume that we have access to a noise-free realization of the underlying random vector. This assumption is not unrealistic on many applications: for example, in image denoising it simply means that we can observe noise-free images that are somewhat similar to the noisy image to be treated, i.e., they belong to the same environment or context. This assumption can be, however, relaxed in many cases, see [7]. The problem of finding an optimal sparse code in step 1 is treated in the next section.

In fact, it turns out that the shrinkage operation given above is quite similar to the one used in the wavelet shrinkage method derived earlier by Donoho et al [3] from a very different approach. Their estimator consisted of applying the shrinkage operator in (6), with different values for the parameters, on the coefficients of the wavelet transform. There are two main differences between the two methods. The first is the choice of the transformation. We choose the transformation using the statistical properties of the data at hand, whereas Donoho et al use a predetermined wavelet transform. The second important difference is that we estimate the shrinkage nonlinearities by the ML principle, again adapting to the data at hand, whereas Donoho et al use fixed thresholding operators derived by the minimax principle.

## 5   Choosing the optimal sparse code

Different measures of sparseness (or nongaussianity) have been proposed in the literature [1, 4, 8, 10]. In this section, we show which measures are optimal for our method. We shall here restrict ourselves to the class of linear, orthogonal transformations. This restriction is justified by the fact that orthogonal transformations leave the gaussian noise structure intact, which makes the problem more simply tractable. This restriction can be relaxed, however, see [7].

A simple, yet very attractive principle for choosing the basis for sparse coding is to consider the data to be generated by a noisy independent component analysis (ICA) model [10, 6, 9]:

$$\mathbf{x} = \mathbf{A}\mathbf{s} + \boldsymbol{\nu}, \tag{9}$$

where the $s_i$ are now the independent components, and $\boldsymbol{\nu}$ is multivariate gaussian noise. We could then estimate $\mathbf{A}$ using ordinary maximum likelihood estimation of the ICA model. Under the restriction that $\mathbf{A}$ is constrained to be orthogonal, estimation of the noise-free components $s_i$ then amounts to the above method of shrinking the values of $\mathbf{A}^T\mathbf{x}$, see [6]. In this ML sense, the optimal transformation matrix is thus given by $\mathbf{W} = \mathbf{A}^T$. In particular, using this principle means that ordinary ICA algorithms can be used to estimate the sparse coding basis. This is very fortunate since the computationally efficient methods for ICA estimation enable the basis estimation even in spaces of rather high dimensions [8, 5].

An alternative principle for determining the optimal sparsifying transformation is to minimize the mean-square error (MSE). In [6], a theorem is given that shows that the optimal basis in minimum MSE sense is obtained by maximizing $\sum_{i=1}^{n} I_F(\mathbf{w}_i^T\mathbf{x})$ where $I_F(s) = E\{[p'(s)/p(s)]^2\}$ is the Fisher information of the density of $s$, and the $\mathbf{w}_i^T$ are the rows of $\mathbf{W}$. Fisher information of a density [4] can be considered as a measure of its nongaussianity. It is well-known [4] that in the set of probability densities of unit variance, Fisher information is minimized by the gaussian density, and the minimum equals 1. Thus the theorem shows that the more nongaussian (sparse) $s$ is, the better we can reduce noise. Note, however, that Fisher information is not scale-invariant.

The former (ML) method of determining the basis matrix gives usually sparser components than the latter method based on minimizing MSE. In the case of image denoising, however, these two methods give essentially equivalent bases if a perceptually weighted MSE is used [6]. Thus we luckily avoid the classical dilemma of choosing between these two optimality criteria.

## 6   Experiments

Image data seems to fulfill the assumptions inherent in sparse code shrinkage: It is possible to find linear representations whose components have sparse distributions, using wavelet-like filters [10]. Thus we performed a set of experiments to explore the utility of sparse code shrinkage in image denoising. The experiments are reported in more detail in [7].

**Data**. The data consisted of real-life images, mainly natural scenes. The images were randomly divided into two sets. The first set was used in estimating the matrix $W$ that gives the sparse coding transformation, as well as in estimating the shrinkage nonlinearities. The second set was used as a test set. It was artificially corrupted by Gaussian noise, and sparse code shrinkage was used to reduce the noise. The images were used in the method in the form of subwindows of $8 \times 8$ pixels.

**Methods**. The sparse coding matrix $W$ was determined by first estimating the ICA model for the image windows (with DC component removed) using the FastICA algorithm [8, 5], and projecting the obtained estimate on the space of orthogonal matrices. The training images were also used to estimate the parametric density models of the sparse components. In the first series of experiments, the local variance was equalized as a preprocessing step [7]. This implied that the density in (5) was a more suitable model for the densities of the sparse components; thus the shrinkage function in (6) was used. In the second series, no such equalization was made, and the density model (7) and the shrinkage function (8) were used [7].

**Results**. Fig. 2 shows, on the left, a test image which was artificially corrupted with Gaussian noise with standard deviation 0.5 (the standard deviations of the original images were normalized to 1). The result of applying our denoising method (without local variance equalization) on that image is shown on the right. Visual comparison of the images in Fig. 2 shows that our sparse code shrinkage method cancels noise quite effectively. One sees that contours and other sharp details are conserved quite well, while the overall reduction of noise is quite strong, which in is contrast to methods based on low-pass filtering. This result is in line with those obtained by wavelet shrinkage [3]. More experimental results are given in [7].

## 7   Conclusion

Sparse coding and ICA can be applied for image feature extraction, resulting in a wavelet-like basis for image windows [10]. As a practical application of such a basis, we introduced the method of sparse code shrinkage. It is based on the fact that in sparse coding the energy of the signal is concentrated on only a few components, which are different for each observed vector. By shrinking the absolute values of the sparse components towards zero, noise can be reduced. The method is also closely connected to modeling image data with noisy independent component analysis [9]. We showed how to find the optimal sparse coding basis for denoising, and we developed families of probability densities that allow the shrinkage nonlinearities to adapt accurately to the data at hand. Experiments on image data showed that the performance of the method is very appealing. The method reduces noise without blurring edges or other sharp features as much as linear low-pass or median filtering. This is made possible by the strongly non-linear nature of the shrinkage operator that takes advantage of the inherent statistical structure of natural images.